# Automated Aircraft Recovery via Reinforcement Learning: Initial Experiments

**Jeffrey F. Monaco**
Barron Associates, Inc.
Jordan Building
1160 Pepsi Place, Suite 300
Charlottesville VA 22901
monaco@bainet.com

**David G. Ward**
Barron Associates, Inc.
Jordan Building
1160 Pepsi Place, Suite 300
Charlottesville VA 22901
ward@bainet.com

**Andrew G. Barto**
Department of Computer Science
University of Massachusetts
Amherst MA 01003
barto@cs.umass.edu

## Abstract

Initial experiments described here were directed toward using reinforcement learning (RL) to develop an automated recovery system (ARS) for high-agility aircraft. An ARS is an outer-loop flight-control system designed to bring an aircraft from a range of out-of-control states to straight-and-level flight in minimum time while satisfying physical and physiological constraints. Here we report on results for a simple version of the problem involving only single-axis (pitch) simulated recoveries. Through simulated control experience using a medium-fidelity aircraft simulation, the RL system approximates an optimal policy for pitch-stick inputs to produce minimum-time transitions to straight-and-level flight in unconstrained cases while avoiding ground-strike. The RL system was also able to adhere to a pilot-station acceleration constraint while executing simulated recoveries.

# 1   INTRODUCTION

An emerging use of reinforcement learning (RL) is to approximate optimal policies for large-scale control problems through extensive simulated control experience.  Described here are initial experiments directed toward the development of an automated recovery system (ARS) for high-agility aircraft. An ARS is an outer-loop flight control system designed to bring the aircraft from a range of initial states to straight, level, and non-inverted flight in minimum time while satisfying constraints such as maintaining altitude and accelerations within acceptable limits. Here we describe the problem and present initial results involving only single-axis (pitch) recoveries. Through extensive simulated control experience using a medium-fidelity simulation of an F-16, the RL system approximated an optimal policy for longitudinal-stick inputs to produce near-minimum-time transitions to straight and level flight in unconstrained cases, as well as while meeting a pilot-station acceleration constraint.

# 2   AIRCRAFT MODEL

The aircraft was modeled as a dynamical system with state vector $x = \{q,\ \alpha,\ p,\ r,\ \beta,\ V_t\}$, where $q$ = body-axes pitch rate, $\alpha$ = angle of attack, $p$ = body-axes roll rate, $r$ = body-axes yaw rate, $\beta$ = angle of sideslip, $V_t$ = total airspeed, and control vector $\delta = \{\delta_{se},\ \delta_{ae},\ \delta_{af},\ \delta_{rud}\}$ of effector and pseudo-effector displacements. The controls are defined as: $\delta_{se}$ = symmetric elevon, $\delta_{ae}$ = asymmetric elevon, $\delta_{af}$ = asymmetric flap, and $\delta_{rud}$ = rudder. (A pseudo-effector is a mathematically convenient combination of real effectors that, e.g., contributes to motion in a limited number of axes.) The following additional descriptive variables were used in the RL problem formulation: $h$ = altitude, $\dot{h}$ = vertical component of velocity, $\Theta$ = pitch attitude, $N_z$ = pilot-station normal acceleration.

For the initial pitch-axis experiment described here, five discrete actions were available to the learning agent in each state; these were longitudinal-stick commands selected from $\{-6, -3, 0, +3, +6\}$ $lbf$. The command chosen by the learning agent was converted into a desired normal-acceleration command through the standard F-16 longitudinal-stick command gradient with software breakout. This gradient maps pounds-of-force inputs into desired acceleration responses. We then produce an approximate relationship between normal acceleration and body-axes pitch rate to yield a pitch-rate flying-qualities model. Given this model, an inner-loop linear-quadratic (LQ) tracking control algorithm determined the actuator commands to result in optimal model-following of the desired pitch-rate response.

The aircraft model consisted of complete translational and rotational dynamics, including nonlinear terms owing to inertial cross-coupling and orientation-dependent gravitational effects. These were obtained from a modified linear F-16 model with dynamics of the form

$$\dot{x} = Ax\ +\ B\delta\ +\ b\ +\ \hat{N}$$

where $A$ and $B$ were the F-16 aero-inertial parameters (stability derivatives) and effector sensitivities (control derivatives). These stability and control derivatives and the bias vector, $b$, were obtained from linearizations of a high-fidelity nonlinear, six-degree-of-freedom model. Nonlinearities owing to inertial cross-coupling and orientation-dependent gravitational effects were accounted for through the term $\hat{N}$, which depended nonlinearly on the state. Nonlinear actuator dynamics were modeled via the incorporation of F-16 effector-rate and effector-position limits. See Ward et al. (1996) for additional details.

# 3   PROBLEM FORMULATION

The RL problem was to approximate a minimum-time control policy capable of bringing the aircraft from a range of initial states to straight, level, and non-inverted flight, while satisfying given constraints, e.g., maintaining the normal acceleration at the pilot station within

an acceptable range. For the single-axis (pitch-axis) flight control problem considered here, recovered flight was defined by:

$$q = \dot{q} = \dot{\alpha} = \dot{h} = \dot{V_t} = 0. \tag{1}$$

Successful recovery was achieved when all conditions in Eq. 1 were satisfied simultaneously within pre-specified tolerances.

Because we wished to distinguish between recovery supplied by the LQ tracker and that learned by the RL system, special attention was given to formulating a meaningful test to avoid falsely attributing successes to the RL system. For example, if initial conditions were specified as off-trim perturbations in body-axes pitch rate, pitch acceleration, and true airspeed, the RL system may not have been required because the LQ controller would provide all the necessary recovery, i.e., zero longitudinal-stick input would result in a commanded body-axes pitch rate of zero *deg./sec.* Because this controller is designed to be highly responsive, its tracking and integral-error penalties usually ensure that the aircraft responses attain the desired state in a relatively short time. The problem was therefore formulated to demand recovery from aircraft orientations where the RL system was primarily responsible for recovery, and the goal state was not readily achieved via the stabilizing action of the LQ control law.

A pitch-axis recovery problem of interest is one in which initial pitch attitude, $\Theta$, is selected to equal $\Theta_{trim} + \mathcal{U}(\hat{\Theta}_{0_{min}}, \hat{\Theta}_{0_{max}})$, where $\Theta_{trim} \equiv \alpha_{trim}$ (by definition), $\mathcal{U}$ is a uniformly distributed random number, and $\Theta_{0_{min}}$ and $\Theta_{0_{max}}$ define the boundaries of the training region, and other variables are set so that when the aircraft is parallel to the earth ($\Theta_0 = 0$), it is "pancaking" toward the ground (with positive trim angle of attack). Other initial conditions correspond to purely-translational climb or descent of the aircraft. For initial conditions where $\Theta_0 < \alpha_{trim}$, the flight vehicle will descend, and in the absence of any corrective longitudinal-stick force, strike the ground or water. Because it imposes no constraints on altitude or pitch-angle variations, the stabilizing response of the LQ controller is inadequate for providing the necessary recovery.

## 4   REINFORCEMENT LEARNING ALGORITHM

Several candidate RL algorithms were evaluated for the ARS. Initial efforts focused primarily on (1) $\mathcal{Q}$-Learning, (2) alternative means for approximating the action-value function ($\mathcal{Q}$ function), and (3) use of discrete versus continuous-action controls. During subsequent investigations, an extension of $\mathcal{Q}$-Learning called Residual Advantage Learning (Baird, 1995; Harmon & Baird, 1996) was implemented and successfully applied to the pitch-axis ARS problem. As with action-values in $\mathcal{Q}$-Learning, the advantage function, $\mathcal{A}(x, u)$, may be represented by a function approximation system of the form

$$\mathcal{A}(x, u) = \phi(x, u)^T \theta, \tag{2}$$

where $\phi(x, u)$ is a vector of relevant features and $\theta$ are the corresponding weights. Here, the advantage function is linear in the weights, $\theta$, and these weights are the modifiable, learned parameters.

For advantage functions of the form in Eq. 2, the update rule is:

$$
\begin{aligned}
\theta_{k+1} &= \theta_k - \alpha \left( \left( (r + \gamma^{\Delta t} \mathcal{A}(y, b^*)) \frac{1}{K\Delta t} + \left(1 - \frac{1}{K\Delta t}\right) \mathcal{A}(x, a^*) - \mathcal{A}(x, a) \right) \right. \\
&\quad \left. \bullet \left( \Phi \gamma^{\Delta t} \phi(y, b^*) \frac{1}{K\Delta t} + \Phi \left(1 - \frac{1}{K\Delta t}\right) \phi(x, a^*) - \phi(x, a) \right) \right),
\end{aligned}
$$

where $a^* = argmin_a \mathcal{A}(x, a)$ and $b^* = argmin_b \mathcal{A}(y, b)$, $\Delta t$ is the system rate (0.02 *sec.* in the ARS), $\gamma^{\Delta t}$ is the discount factor, and $K$ is an fixed scale factor. In the above notation,

$y$ is the resultant state, i.e., the execution of action $a$ results in a transition from state $x$ to its successor $y$.

The Residual Advantage Learning update collapses to the $Q$-Learning update for the case $\Phi = 0$, $K = \frac{1}{\Delta t}$. The parameter $\Phi$ is a scalar that controls the trade-off between residual-gradient descent when $\Phi = 1$, and a faster, direct algorithm when $\Phi = 0$. Harmon & Baird (1996) address the choice of $\Phi$, suggesting the following computation of $\Phi$ at each time step:

$$\Phi = \frac{\sum_\theta w_d w_{rg}}{\sum_\theta (w_d - w_{rg})w_{rg}} + \mu$$

where $w_d$ and $w_{rg}$ are *traces* (one for each $\theta$ of the function approximation system) associated with the direct and residual gradient algorithms, respectively, and $\mu$ is a small, positive constant that dictates how rapidly the system forgets. The traces are updated during each cycle as follows

$$w_d \leftarrow (1 - \mu)w_d - \mu \left[ (r + \gamma^{\Delta t}\mathcal{A}(y, b^*)) \frac{1}{K\Delta t} + \left( 1 - \frac{1}{K\Delta t} \right) \mathcal{A}(x, a^*) \right]$$
$$\bullet \left[ -\frac{\partial}{\partial \theta} \mathcal{A}(x, a^*) \right]$$

$$w_{rg} \leftarrow (1 - \mu)w_{rg} - \mu \left[ (r + \gamma^{\Delta t}\mathcal{A}(y, b^*)) \frac{1}{K\Delta t} + (1 - \frac{1}{K\Delta t})\mathcal{A}(x, a^*) - \mathcal{A}(x, a) \right]$$
$$\bullet \left[ \gamma^{\Delta t} \frac{\partial}{\partial \theta} \mathcal{A}(y, b^*) \frac{1}{K\Delta t} + (1 - \frac{1}{K\Delta t}) \frac{\partial}{\partial \theta} \mathcal{A}(x, a^*) - \frac{\partial}{\partial \theta} \mathcal{A}(x, a) \right].$$

Advantage Learning updates of the weights, including the calculation of an adaptive $\Phi$ as discussed above, were implemented and interfaced with the aircraft simulation. The Advantage Learning algorithm consistently outperformed its $Q$-Learning counterpart. For this reason, most of our efforts have focused on the application of Advantage Learning to the solution of the ARS. The feature vector $\phi(x, u)$ consisted of normalized (dimensionless) states and controls, and functions of these variables. Use of these nondimensionalized variables (obtained via the Buckingham $\pi$-theorem; e.g., Langharr, 1951) was found to enhance greatly the stability and robustness of the learning process. Furthermore, the RL system appeared to be less sensitive to changes in parameters such as the learning rate when these techniques were employed.

# 5 TRAINING

Training the RL system for arbitrary orientations was accomplished by choosing random initial conditions on $\Theta$ as outlined above. With the exception of $h$, all other initial conditions corresponded to trim values for a Mach 0.6, 5 $kft$. flight condition. Rewards were $-1$ per-time-step until the goal state was reached. In preliminary experiments, the training region was restricted to $\pm 0.174$ $rad.(10$ $deg.)$ from the trim pitch angle. For this range of initial conditions, the system was able to learn an appropriate policy given only a handful of features (approximately 30). The policy was significantly mature after 24 hours of learning on an HP-730 workstation and appeared to be able to achieve the goal for arbitrary initial conditions in the aforementioned domain.

We then expanded the training region and considered initial $\Theta$ values within $\pm 0.785$ $rad.$ $(45$ $deg.)$ of trim. The policy previously learned for the more restricted training domain performed well here too, and learning to recover for these more drastic off-trim conditions was trivial. No boundary restrictions were imposed on the system, but a report of whether the aircraft would have struck the ground was maintained. It was noted

that recovery from all possible initial conditions could not be achieved without hitting the ground. Episodes in which the ground would have been encountered were a result of inadequate control authority and not an inadequate RL policy. For example, when the initial pitch angle was at its maximum negative value, maximum-allowable positive stick (6 *lbf.*) was not sufficient to pull up the aircraft nose in time. To remedy this in subsequent experiments, the number of admissible actions was increased to include larger-magnitude commands: $\{-12, -9, -6, -3, 0, +3, +6, +9, +12\}$ *lbf.*

Early attempts at solving the pitch-axis recovery problem with the expanded initial conditions in conjunction with this augmented action set proved challenging. The policy that worked well in the two previous experiments was no longer able to attain the goal state; it was only able to come close and oscillate indefinitely about the goal region. The agent learned to pitch up and down appropriately, e.g., when $\dot{h}$ was negative it applied a corrective positive action, and *vice versa*. However, because of system and actuator dynamics modeled in the simulation, the transient response caused the aircraft to pass through the goal state. Once beyond the goal region, the agent applied an opposite action, causing it to approach the goal state again, repeating the process indefinitely (until the system was reset and a new trial was started). Thus, the availability of large-amplitude commands and the presence of actuator dynamics made it difficult for the agent to formulate a consistent policy that afforded all goal state criteria being satisfied simultaneously. One might remedy the problem by removing the actuator dynamics; however, we did not wish to compromise simulation fidelity, and chose to use an expanded feature set to improve RL performance. Using a larger collection of features with approximately 180 inputs, the RL agent was able to formulate a consistent recovery policy. The learning process required approximately 72 hours on an HP-730 workstation. (On this platform, the combined aircraft simulation and RL software execution rate was approximately twice that of real-time.) At this point performance was evaluated. The simulation was run in evaluation mode, i.e., learning rate was set to zero and random exploration was disabled. Performance is summarized below.

## 6   RESULTS

### 6.1   UNCONSTRAINED PITCH-AXIS RECOVERY

Fig. 1 shows the transition times from off-trim orientations to the goal state as a function of initial pitch (inclination) angle. Recovery times were approximately 11–12 *sec.* for the worst-case scenarios, i.e., $|\Theta_0| = 45$ *deg.* off-trim, and decrease (almost) monotonically for points closer to the unperturbed initial conditions. The occasional "blips" in the figure suggest that additional learning would have improved the global RL performance slightly. For $|\Theta_0| = 45$ *deg.* off-trim, maximum altitude loss and gain were each approximately 1667 *ft.* ($0.33 \times 5000 ft.$). These excursions may seem substantial, but when one looks at the time histories for these maneuvers, it is apparent that the RL-derived policy was performing well. The policy effectively minimizes any altitude variation; the magnitude of these changes are principally governed by available control authority and the severity of the flight condition from which the policy must recover.

Fig. 2 shows time histories of relevant variables for one of the limiting cases. The first column shows body-axes pitch rate ($Qb$) and commanded body-axes pitch rate ($Qbmodel$) in ($deg./sec.$), pilot station normal acceleration ($Nz$) in ($g$), angle of attack ($Alpha$) in ($deg.$), and pitch attitude ($Theta$) in ($deg.$), respectively. The second column shows the longitudinal stick action executed by the RL system ($lbf.$), the left and right elevator deflections ($deg.$), total airspeed ($ft./sec.$), and altitude ($ft.$). The majority of the 1600+ *ft.* altitude loss occurs between zero and five sec.; during this time, the RL system is applying maximum (allowable) positive stick. Thus, this altitude excursion is principally attributed to limited control authority as well as significant off-trim initial orientations.

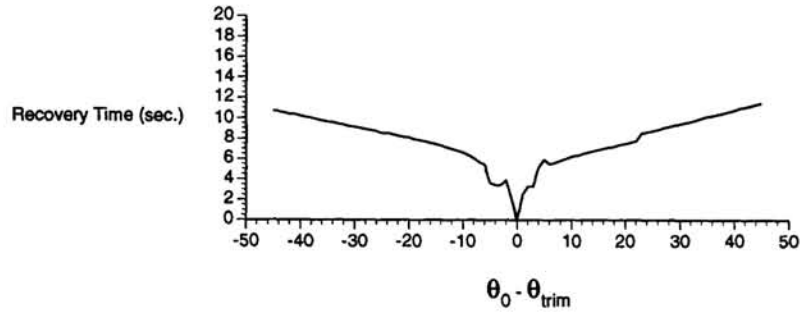

Figure 1: Simulated Aircraft Recovery Times for Unconstrained Pitch-Axis ARS

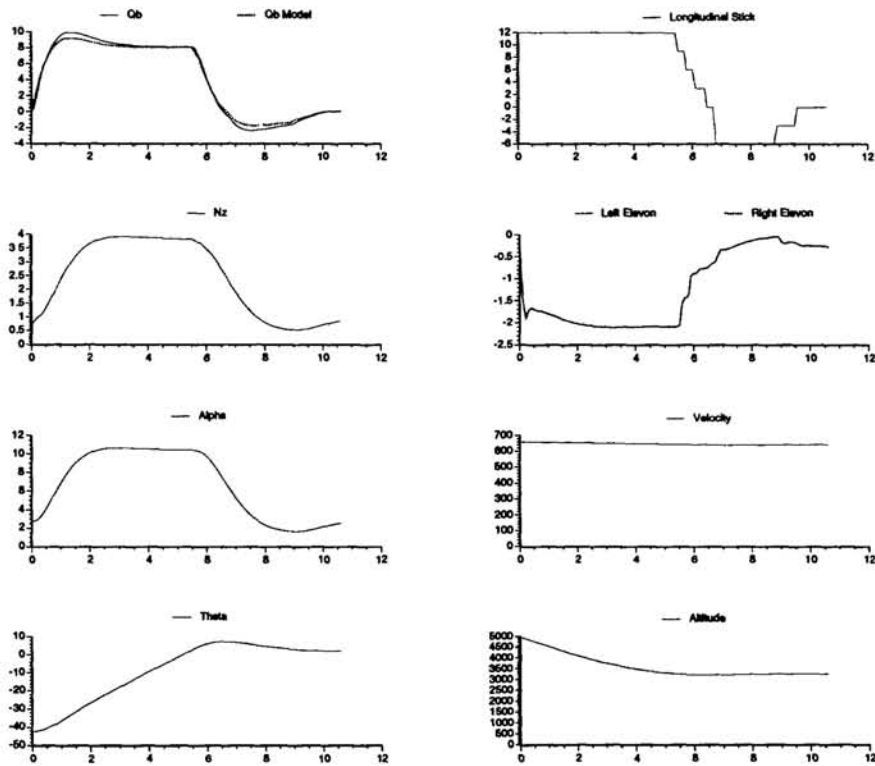

Figure 2: Time Histories During Unconstrained Pitch-Axis Recovery for $\Theta_0 = \Theta_{trim} - 45\,deg.$

## 6.2  CONSTRAINED PITCH-AXIS RECOVERY

The requirement to execute aircraft recoveries while adhering to pilot-safety constraints was a deciding factor in using RL to demonstrate the automated recovery system concept. The need to recover an aircraft while minimizing injury and, where possible, discomfort to the flight crew, requires that the controller incorporate constraints that can be difficult or impossible to express in forms suitable for linear and nonlinear programming methods.

In subsequent ARS investigations, allowable pilot-station normal acceleration was restricted to the range $-1.5\,g \leq N_z \leq 3.5\,g$. These values were selected because the unconstrained ARS was observed to exceed these limits. Several additional features (for a total of 189) were chosen, and the learning process was continued. Initial weights for the original 180 inputs corresponded to those from the previously learned policy; the new features were chosen to have zero weights initially. Here, the RL system learned to avoid the normal acceleration limits and consistently reach the goal state for initial pitch angles in the region $[-45 + \Theta_{trim}, 35 + \Theta_{trim}]$ $deg$. Additional learning should result in improved recovery policies in this bounded acceleration domain for all initial conditions. Nonetheless, the results showed how an RL system can learn to satisfy these kinds of constraints.

## 7  CONCLUSION

In addition to the results reported here, we conducted extensive analysis of the degree to which the learned policy successfully generalized to a range of initial conditions not experienced in training. In all cases, aircraft responses to novel recovery scenarios were stable and qualitatively similar to those previously executed in the training region. We are also conducting experiments with a multi-axes ARS, in which longitudinal-stick and lateral-stick sequences must be coordinated to recover the aircraft. Initial results are promising, but substantially longer training times are required. In summary, we believe that the results presented here demonstrate the feasibility of using RL algorithms to develop robust recovery strategies for high-agility aircraft, although substantial further research is needed.

### Acknowledgments

This work was supported by the Naval Air Warfare Center Aircraft Division (NAWCAD), Flight Controls/Aeromechanics Division under Contract N62269-96-C-0080. The authors thank Marc Steinberg, the Program Manager and Chief Technical Monitor. The authors also express appreciation to Rich Sutton and Mance Harmon for their valuable help, and to Lockheed Martin Tactical Aircraft Systems for authorization to use their *ATLAS* software, from which F-16 parameters were extracted.

### References

Baird, L. C. (1995) Residual algorithms: reinforcement learning with function approximation. In A. Prieditis and S. Russell (eds.), *Machine Learning: Proceedings of the Twelfth International Conference*, pp. 30-37. San Francisco, CA: Morgan Kaufmann.

Harmon, M. E. & Baird, L. C. (1996) Multi-agent residual advantage learning with general function approximation. Wright Laboratory Technical Report, WPAFB, OH.

Langharr, H. L. (1951) *Dimensional Analysis and Theory of Models*. New York: Wiley and Sons.

Ward, D. G., Monaco, J. F., Barron, R. L., Bird, R.A., Virnig, J.C., & Landers, T.F. (1996) Self-designing controller. Final Tech. Rep. for Directorate of Mathematics and Computer Sciences, AFOSR, Contract F49620-94-C-0087. Barron Associates, Inc.